# One-Class LP Classifier for Dissimilarity Representations

Elżbieta Pękalska[1], David M.J.Tax[2] and Robert P.W. Duin[1]

[1]Delft University of Technology, Lorentzweg 1, 2628 CJ Delft, The Netherlands
[2]Fraunhofer Institute FIRST.IDA, Kekuléstr.7, D-12489 Berlin, Germany
ela@ph.tn.tudelft.nl,davidt@first.fraunhofer.de

## Abstract

Problems in which abnormal or novel situations should be detected can be approached by describing the domain of the class of typical examples. These applications come from the areas of machine diagnostics, fault detection, illness identification or, in principle, refer to any problem where little knowledge is available outside the typical class. In this paper we explain why proximities are natural representations for domain descriptors and we propose a simple one-class classifier for dissimilarity representations. By the use of linear programming an efficient one-class description can be found, based on a small number of prototype objects. This classifier can be made (1) more robust by transforming the dissimilarities and (2) cheaper to compute by using a reduced representation set. Finally, a comparison to a comparable one-class classifier by Campbell and Bennett is given.

## 1 Introduction

The problem of describing a class or a domain has recently gained a lot of attention, since it can be identified in many applications. The area of interest covers all the problems, where the specified targets have to be recognized and the anomalies or outlier instances have to be detected. Those might be examples of any type of fault detection, abnormal behavior, rare illnesses, etc. One possible approach to class description problems is to construct one-class classifiers (OCCs) [13]. Such classifiers are concept descriptors, i.e. they refer to all possible knowledge that one has about the class.

An efficient OCC built in a feature space can be found by determining a minimal volume hypersphere around the data [14, 13] or by determining a hyperplane such that it separates the data from the origin as well as possible [11, 12]. By the use of kernels [15] the data is implicitly mapped into a higher-dimensional inner product space and, as a result, an OCC in the original space can yield a nonlinear and non-spherical boundary; see e.g. [15, 11, 12, 14].

Those approaches are convenient for data already represented in a feature space. In some cases, there is, however, a lack of good or suitable features due to the difficulty of defining them, as e.g. in case of strings, graphs or shapes. To avoid the definition of an explicit feature space, we have already proposed to address kernels as general proximity measures [10] and not only as symmetric, (conditionally) positive definite functions of two variables

[2]. Such a proximity should directly arise from an application; see e.g. [8, 7]. Therefore, our reasoning starts *not* from a feature space, like in case of the other methods [15, 11, 12, 14], but from a given proximity representation. Here, we address general dissimilarities.

The basic assumption that an instance belongs to a class is that it is similar to examples within this class. The identification procedure is realized by a proximity function equipped with a threshold, determining whether an instance is a class member or not. This proximity function can be e.g. a distance to an average representative, or a set of selected proto-types. The data represented by proximities is thus more natural for building the concept descriptors, i.e. OCCs, since the proximity function can be directly built on them.

In this paper, we propose a simple and efficient OCC for general dissimilarity represen-tations, discussed in Section 2, found by the use of linear programming (LP). Section 3 presents our method together with a dissimilarity transformation to make it more robust against objects with large dissimilarities. Section 4 describes the experiments conducted, and discusses the results. Conclusions are summarized in Section 5.

## 2 Dissimilarity representations

Although a dissimilarity measure $D$ provides a flexible way to represent the data, there are some constraints. Reflectivity and positivity conditions are essential to define a proper measure; see also [10]. For our convenience, we also adopt the symmetry requirement. We do not require that $D$ is a strict metric, since non-metric dissimilarities may naturally be found when shapes or objects in images are compared e.g. in computer vision [4, 7]. Let $z$ and $p_i$ refer to objects to be compared. A dissimilarity representation can now be seen as a dissimilarity kernel based on the representation set $R = \{p_1, .., p_N\}$ and realized by a mapping $D(z, R) : \mathcal{F} \to \mathcal{R}^N$, defined as $D(z, R) = [D(z, p_1) \ldots D(z, p_N)]^T$. $R$ controls the dimensionality of a dissimilarity space $D(\cdot, R)$. Note also that $\mathcal{F}$ expresses a conceptual space of objects, not necessarily a feature space. Therefore, to emphasize that objects, like $z$ or $p_i$, might not be feature vectors, they will *not* be printed in bold.

The compactness hypothesis (CH) [5] is the basis for object recognition. It states that similar objects are close in their representations. For a dissimilarity measure $D$, this means that $D(r, s)$ is small if objects $r$ and $s$ are similar. If we demand that $D(r, s) = 0$, if and only if the objects $r$ and $s$ are identical, this implies that they belong to the same class. This can be extended by assuming that all objects $s$ such that $D(r, s) < \varepsilon$, for a sufficient small $\varepsilon$, are so similar to $r$ that they are members of the same class. Consequently, $D(r, t) \approx D(s, t)$ for other objects $t$. Therefore, for dissimilarity representations satisfying the above continuity, the reverse of the CH holds: objects similar in their representations are similar in reality and belong, thereby, to the same class [6, 10].

Objects with large distances are assumed to be dissimilar. When the set $R$ contains objects from the class of interest, then objects $z$ with large $D(z, R)$ are outliers and should be remote from the origin in this dissimilarity space. This characteristic will be used in our OCC. If the dissimilarity measure $D$ is a metric, then all vectors $D(z, R)$, lie in an open prism (unbounded from above[1]), bounded from below by a hyperplane on which the objects from $R$ are. In principle, $z$ may be placed anywhere in the dissimilarity space $D(\cdot, R)$ only if the triangle inequality is completely violated. This is, however, not possible from the practical point of view, because then both the CH and its reverse will not be fulfilled. Consequently, this would mean that $D$ has lost its discriminating properties of being small for similar objects. Therefore, the measure $D$, if not a metric, has to be only slightly non-metric (i.e. the triangle inequalities are only somewhat violated) and, thereby, $D(z, R)$ will still lie either in the prism or in its close neigbourhood.

## 3 The linear programming dissimilarity data description

To describe a class in a non-negative dissimilarity space, one could minimize the volume of the prism, cut by a hyperplane $P\colon \boldsymbol{w}^T D(z, R) = \rho$ that bounds the data from above[2] (note that non-negative dissimilarities impose both $\rho \geq 0$ and $w_i \geq 0$). However, this might be not a feasible task. A natural extension is to minimize the volume of a simplex with the main vertex being the origin and the other vertices $\boldsymbol{v}_j$ resulting from the intersection of $P$ and the axes of the dissimilarity space ($\boldsymbol{v}_j$ is a vector of all zero elements except for $v_{ji} = \rho/w_i$, given that $w_i \neq 0$). Assume now that there are $M$ non-zero weights of the hyperplane $P$, so effectively, $P$ is constructed in a $\mathcal{R}^M$. From geometry we know that the volume $V$ of such a simplex can be expressed as $V = (V_{\text{Base}}/M!) \cdot (\rho/||\boldsymbol{w}||_2)$, where $V_{\text{Base}}$ is the volume of the base, defined by the vertices $\boldsymbol{v}_j$. The minimization of $h = \rho/||\boldsymbol{w}||_2$, i.e. the Euclidean distance from the origin to $P$, is then related to the minimization of $V$.

Let $\{D(p_i, R)\}_{i=1}^N, N = |R|$ be a dissimilarity representation, bounded by a hyperplane $P$, i.e. $\boldsymbol{w}^T D(p_i, R) \leq \rho$ for $i = 1, \ldots, N$, such that the $L_q$ distance to the origin $d_q(\boldsymbol{0}, P) = \rho/||\boldsymbol{w}||_p$ is the smallest (i.e. $q$ satisfies $1/p + 1/q = 1$ for $p \geq 1$) [9]. This means that $P$ can be determined by minimizing $\rho - ||\boldsymbol{w}||_p$. However, when we require $||\boldsymbol{w}||_p = 1$ (to avoid any arbitrary scaling of $\boldsymbol{w}$), the construction of $P$ can be solved by the minimization of $\rho$ only. The mathematical programming formulation of such a problem is [9, 1]:

$$\begin{aligned} \min \quad & \rho \\ \text{s.t.} \quad & \boldsymbol{w}^T D(p_i, R) \leq \rho, \qquad i = 1, 2, .., N, \qquad ||\boldsymbol{w}||_p = 1, \ \rho \geq 0. \end{aligned} \tag{1}$$

If $p = 2$, then $P$ is found such that $h$ is minimized, yielding a quadratic optimization problem. A much simpler LP formulation, realized for $p = 1$, is of our interest. Knowing that $||\boldsymbol{w}||_2 \leq ||\boldsymbol{w}||_1 \leq \sqrt{M}||\boldsymbol{w}||_2$ and by the assumption of $||\boldsymbol{w}||_1 = 1$, after simple calculations, we find that $\rho \leq h = \rho/||\boldsymbol{w}||_2 \leq \sqrt{M}\rho$. Therefore, by minimizing $d_\infty(\boldsymbol{0}, P) = \rho$, (and $||\boldsymbol{w}||_1 = 1$), $h$ will be bounded and the volume of the simplex considered, as well.

By the above reasoning and (1), a class represented by dissimilarities can be characterized by a linear proximity function with the weights $\boldsymbol{w}$ and the threshold $\rho$. Our one-class classifier $\mathcal{C}_{\text{LPDD}}$, Linear Programming Dissimilarity-data Description, is then defined as:

$$\mathcal{C}_{\text{LPDD}}(D(z, \cdot)) = \mathcal{I}(\sum_{w_j \neq 0} w_j D(z, p_j) \leq \rho), \tag{2}$$

where $\mathcal{I}$ is the indicator function. The proximity function is found as the solution to a soft margin formulation (which is a straightforward extension of the hard margin case) with $\nu \in (0, 1]$ being the upper bound on the outlier fraction for the target class:

$$\begin{aligned} \min \quad & \rho + \frac{1}{\nu N} \sum_{i=1}^N \xi_i \\ \text{s.t.} \quad & \boldsymbol{w}^T D(p_i, R) \leq \rho + \xi_i, \quad i = 1, 2, .., N \\ & \sum_j w_j = 1, \ w_j \geq 0, \ \rho \geq 0, \ \xi_i \geq 0. \end{aligned} \tag{3}$$

In the LP formulations, sparse solutions are obtained, meaning that only some $w_j$ are positive. Objects corresponding to such non-zero weights, will be called *support objects* (SO).

The left plot of Fig. 1 is a 2D illustration of the LPDD. The data is represented in a metric dissimilarity space, and by the triangle inequality the data can only be inside the prism indicated by the dashed lines. The LPDD boundary is given by the hyperplane, as close to the origin as possible (by minimizing $\rho$), while still accepting (most) target objects. By the discussion in Section 2, the outliers should be remote from the origin.

**Proposition.** In (3), $\nu \in (0, 1]$ is the upper bound on the outlier fraction for the target class, i.e. the fraction of objects that lie outside the boundary; see also [11, 12]. This means that $\frac{1}{N} \sum_{i=1}^N (1 - \mathcal{C}_{\text{LPDD}}(D(p_i, \cdot))) \leq \nu$.

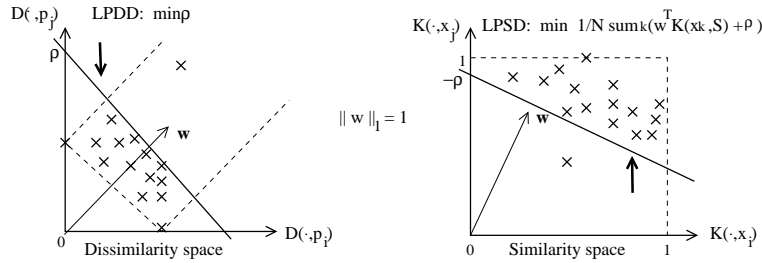

Figure 1: Illustrations of the LPDD in the dissimilarity space (left) and the LPSD in the similarity space (right). The dashed lines indicate the boundary of the area which contains the genuine objects. The LPDD tries to minimize the max-norm distance from the bounding hyperplane to the origin, while the LPSD tries to attract the hyperplane towards the average of the distribution.

The proof goes analogously to the proofs given in [11, 12]. Intuitively, the proof follows this: assume we have found a solution of (3). If $\rho$ is increased slightly, the term $\sum_i \xi_i$ in the objective function will change proportionally to the *number* of points that have non-zero $\xi_i$ (i.e. the outlier objects). At the optimum of (3) it has to hold that $N\nu \geq \#$outliers.

**Scaling dissimilarities.** If $D$ is unbounded, then some atypical objects of the target class (i.e. with large dissimilarities) might badly influence the solution of (3). Therefore, we propose a nonlinear, monotonous transformation of the distances to the interval $[0, 1]$ such that locally the distances are scaled linearly and globally, all large distances become close to 1. A function with such properties is the sigmoid function (the hyperbolical tangent can also be used), i.e. $\mathrm{Sigm}(x) = 2/(1 + e^{-x/s}) - 1$, where $s$ controls the 'slope' of the function, i.e. the size of the local neighborhoods. Now, the transformation can be applied in an element-wise way to the dissimilarity representation such that $D_s(z, p_i) = \mathrm{Sigm}(D(z, p_i))$. Unless stated otherwise, the $\mathcal{C}_{\mathrm{LPDD}}$ will be trained on $D_s$.

**A linear programming OCC on similarities**. Recently, Campbell and Bennett have proposed an LP formulation for novelty detection [3]. They start their reasoning from a feature space in the spirit of positive definite kernels $K(S, S)$ based on the set $S = \{\boldsymbol{x}_1, .., \boldsymbol{x}_N\}$. They restricted themselves to the (modified) RBF kernels, i.e. for $K(\boldsymbol{x}_i, \boldsymbol{x}_j) = e^{-D(\boldsymbol{x}_i, \boldsymbol{x}_j)^2/2\,s^2}$, where $D$ is either Euclidean or $L_1$ (city block) distance. In principle, we will refer to $\mathrm{RBF}_p$, as to the 'Gaussian' kernel based on the $L_p$ distance. Here, to be consistent with our LPDD method, we rewrite their soft-margin LP formulation (a hard margin formulation is then obvious), to include a trade-off parameter $\nu$ (which lacks, however, the interpretation as given in the LPDD), as follows:

$$
\begin{aligned}
\min \quad & \tfrac{1}{N}\sum_{i=1}^{N}(\boldsymbol{w}^T K(\boldsymbol{x}_i, S) + \rho) + \tfrac{1}{\nu N}\sum_{i=1}^{N}\xi_i \\
\text{s.t.} \quad & \boldsymbol{w}^T K(\boldsymbol{x}_i, S) + \rho \geq -\xi_i, \quad i = 1, 2, .., N \\
& \sum_j w_j = 1, \; w_j \geq 0, \; \xi_i \geq 0.
\end{aligned}
\tag{4}
$$

Since $K$ can be any similarity representation, for simplicity, we will call this method Linear Programming Similarity-data Description (LPSD). The $\mathcal{C}_{\mathrm{LPSD}}$ is then defined as:

$$
\mathcal{C}_{\mathrm{LPSD}}(K(\boldsymbol{z}, \cdot)) = \mathcal{I}(\sum_{w_j \neq 0} w_j K(\boldsymbol{z}, \boldsymbol{x}_j) + \rho \geq 0).
\tag{5}
$$

In the right plot of Fig. 1, a 2D illustration of the LPSD is shown. Here, the data is represented in a similarity space, such that all objects lie in a hypercube between 0 and 1. Objects remote from the representation objects will be close to the origin. The hyperplane is optimized to have minimal *average* output for the whole target set. This does not necessarily mean a good separation from the origin or a small volume of the OCC, possibly resulting in an unnecessarily high outlier acceptance rate.

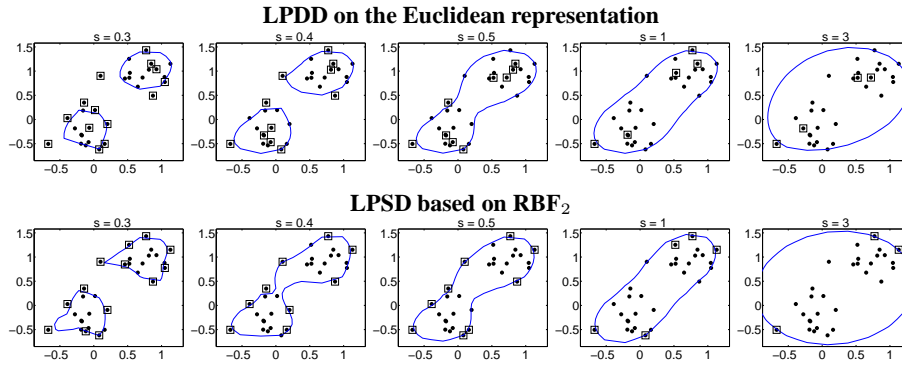

**LPDD on the Euclidean representation**

**LPSD based on RBF$_2$**

Figure 2: One-class hard margin LP classifiers for an artificial 2D data. From left to right, $s$ takes the values of $0.3d, 0.4d, 0.5d, d, 3d$, where $d$ is the average distance. Support objects are marked by squares.

**Extensions**. Until now, the LPDD and LPSD were defined for square (dis)similarity matrices. If the computation of (dis)similarities is very costly, one can consider a reduced representation set $R_{\mathrm{red}} \subset R$, consisting of $n \ll N$ objects. Then, a dissimilarity or similarity representations are given as rectangular matrices $D(R, R_{\mathrm{red}})$ or $K(S, S_{\mathrm{red}})$, respectively. Both formulations (3) and (4) remain the same with the only change that $R/S$ is replaced by $R_{\mathrm{red}}/S_{\mathrm{red}}$. An another reason to consider reduced representations is the robustness against outliers. How to choose such a set is beyond the scope of this paper.

## 4   Experiments

**Artificial datasets**. First, we illustrate the LPDD and the LPSD methods on two artificial datasets, both originally created in a 2D feature space. The first dataset contains two clusters with objects represented by Euclidean distances. The second dataset contains one uniform, square cluster and it is contaminated with three outliers. The objects are represented by a slightly non-metric $L_{0.95}$ dissimilarity (i.e. $d_{0.95}(\boldsymbol{x}, \boldsymbol{y}) = [\sum_i (x_i - y_i)^{0.95}]^{1/0.95}$). In Fig. 2, the first dataset together with the decision boundaries of the LPDD and the LPSD in the theoretical input space are shown. The parameter $s$ used in all plots refers either to the scaling parameter in the sigmoid function for the LPDD (based on $D_s$) or to the scaling parameter in the RBF kernel. The pictures show similar behavior of both the LPDD and the LPSD; the LPDD tends to be just slightly more tight around the target class.

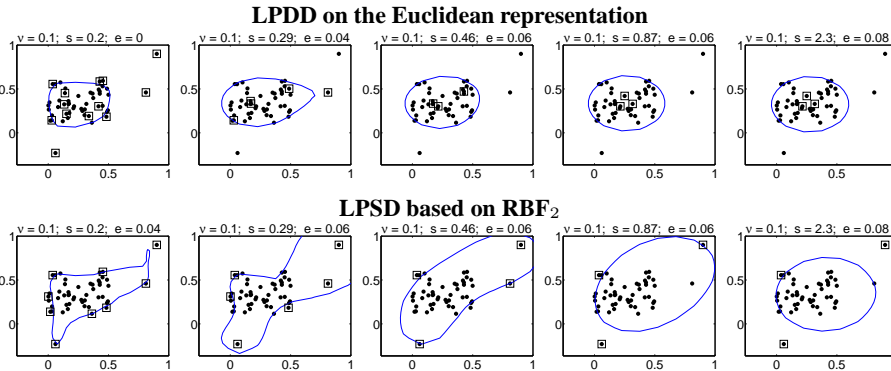

**LPDD on the Euclidean representation**

**LPSD based on RBF$_2$**

Figure 3: One-class LP classifiers, trained with $\nu = 0.1$ for an artificial uniformly distributed 2D data with 3 outliers. From left to right $s$ takes the values of $0.7d_m, d_m, 1.6d_m, 3d_m, 8d_m$, where $d_m$ is the median distance of all the distances. $e$ refers to the error on the target set. Support objects are marked by squares.

**LPDD on the $L_{0.95}$ representation**

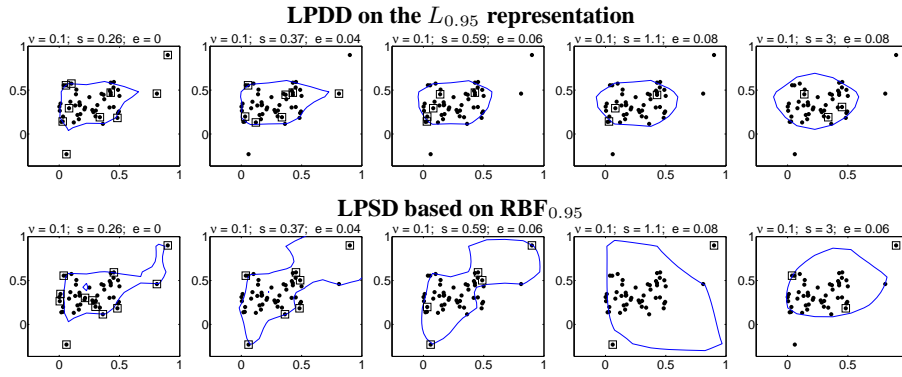

Figure 4: One-class LP classifiers for an artificial 2D data. The same setting as in Fig.3 is used, only for the $L_{0.95}$ non-metric dissimilarities instead of the Euclidean ones. Note that the median distance has changed, and consequently, the $s$ values, as well.

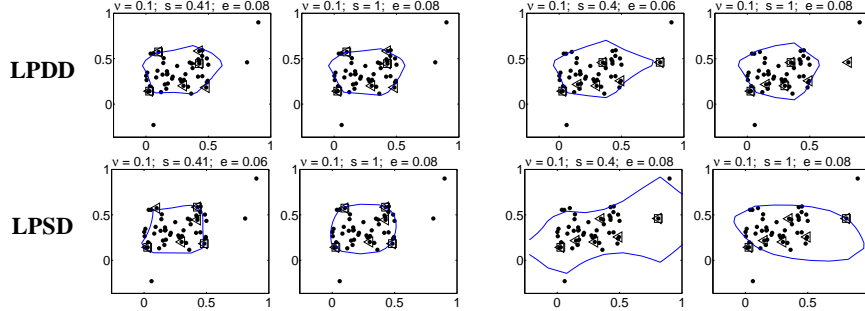

Figure 5: One-class LP classifiers, trained with $\nu = 0.1$, for an artificial uniformly distributed 2D data with 3 outliers, given by the $L_{0.95}$ non-metric *rectangular* $50 \times 6$ dissimilarity representations. The upper row shows the LPDD's results and bottom row shows the LPSD's results with the kernel RBF$_{0.95}$. The objects of the reduced sets $R_{red}$ and $S_{red}$ are marked by triangles. Note that they differ from left to right. $e$ refers to the error on the target set. Support objects are marked by squares.

This becomes more clear in Fig. 3 and 4, where three outliers lying outside a single uniformly distributed cluster should be ignored when an OCC with a soft margin is trained. From these figures, we can observe that the LPDD gives a tighter class description, which is more robust against the scaling parameter and more robust against outliers, as well. The same is observed when $L_{0.95}$ dissimilarity is used instead of the Euclidean distances.

Fig. 5 presents some results for the reduced representations, in which just 6 objects are randomly chosen for the set $R_{red}$. In the left four plots, $R_{red}$ contains objects from the uniform cluster only, and both methods perform equally well. In the right four plots, on the other hand, $R_{red}$ contains an outlier. It can be judged that for a suitable scaling $s$, no outliers become support objects in the LPDD, which is often a case for the LPSD; see also Fig. 4 and 3. Also, a crucial difference between the LPDD and LPSD can be observed w.r.t. the support objects. In case of the LPSD (applied to a non-reduced representation), they lie on the boundary, while in case of the LPDD, they tend to be 'inside' the class.

**Condition monitoring**. Fault detection is an important problem in the machine diagnostics: failure to detect faults can lead to machine damage, while false alarms can lead to unnecessary expenses. As an example, we will consider a detection of four types of fault in ball-bearing cages, a dataset [16] considered in [3]. Each data instance consists of 2048 samples of acceleration taken with a Bruel and Kjaer vibration analyser. After pre-processing with a discrete Fast Fourier Transform, each signal is characterized by 32 attributes. The dataset consists of five categories: normal behavior (NB), corresponding

Table 1: The errors of the first and second kind (in %) of the LPDD and LPSD on two dissimilarity representations for the ball-bearing data. The reduced representations are based on 180 objects.

| Euclidean representation | | | | | | | |
|---|---|---|---|---|---|---|---|
| Method | | | Error | | | | |
| | Optimal $s$ | # of SO | NB | $T_1$ | $T_2$ | $T_3$ | $T_4$ |
| LPDD | 200.4 | 10 | 1.4 | 0.0 | 45.0 | 69.8 | 70.0 |
| LPDD-reduced | 65.3 | 17 | 1.1 | 0.0 | 20.2 | 47.5 | 50.9 |
| LPSD | 320.0 | 8 | 1.3 | 0.0 | 46.7 | 71.7 | 74.5 |
| LPSD-reduced | 211.2 | 6 | 0.6 | 0.0 | 39.9 | 67.1 | 69.5 |
| $L_1$ dissimilarity representation | | | | | | | |
| Method | | | Error | | | | |
| | Optimal $s$ | # of SO | NB | $T_1$ | $T_2$ | $T_3$ | $T_4$ |
| LPDD | 566.3 | 12 | 1.3 | 0.0 | 1.6 | 20.9 | 19.8 |
| LPDD-reduced | 329.5 | 10 | 1.3 | 0.0 | 2.3 | 18.7 | 16.9 |
| LPSD | 1019.3 | 8 | 0.9 | 0.0 | 2.2 | 27.9 | 27.2 |
| LPSD-reduced | 965.7 | 5 | 0.3 | 0.0 | 3.5 | 26.3 | 27.5 |

to measurements made from new ball-bearings, and four types of anomalies, say, $T_1 - T_4$, corresponding either to the damaged outer race or cages or a badly worn ball-bearing. To compare our LPDD method with the LPSD method, we performed experiments in the same way, as described in [3], making use of the same training set, and independent validation and test sets; see Fig. 6.

The optimal values of $s$ were found for both LPDD and LPSD methods by the use of the validation set on the Euclidean and $L_1$ dissimilarity representations. The results are presented in Table 1. It can be concluded that the $L_1$ representation is far more convenient for the fault detection, especially if we look at the fault type $T_3$ and $T_4$ which were unseen in the validation process. The LPSD performs better on normal instances (yields a smaller error) than the LPDD. This is to be expected, since the

| | Train | Valid. | Test |
|---|---|---|---|
| NB | 913 | 913 | 913 |
| $T_1$ | | 747 | 747 |
| $T_2$ | | 913 | 996 |
| $T_3$ | | | 996 |
| $T_4$ | | | 996 |

Figure 6: Fault detection data.

boundary is less tight, by which less support objects (SO) are needed. On the contrary, the LPSD method deteriorates w.r.t. the outlier detection. Note also that the reduced representation, based on randomly chosen 180 target objects ($\approx 20\%$) mostly yields significantly better performances in outlier detection for the LPDD, and in target acceptance for the LPSD. Therefore, we can conclude that if a failure in the fault detection has higher costs than the cost of misclassifying target objects, then our approach should be recommended.

## 5 Conclusions

We have proposed the Linear Programming Dissimilarity-data Description (LPDD) classifier, directly built on dissimilarity representations. This method is efficient, which means that only some objects are needed for the computation of dissimilarities in a test phase. The novelty of our approach lies in its reformulation for general dissimilarity measures, which, we think, is more natural for class descriptors. Since dissimilarity measures might be unbounded, we have also proposed to transform dissimilarities by the sigmoid function, which makes the LPDD more robust against objects with large dissimilarities. We emphasized the possibility of using the LP procedures for rectangular dissimilarity/similarity representations, which is especially useful when (dis)similarities are costly to compute.

The LPDD is applied to artificial and real-world datasets and compared to the LPSD detector as proposed in [3]. For all considered datasets, the LPDD yields a more compact target description than the LPSD. The LPDD is more robust against outliers in the training set, in

particular, when only some objects are considered for a reduced representation. Moreover, with a proper scaling parameter $s$ of the sigmoid function, the support objects in the LPDD do not contain outliers, while it seems difficult for the LPSD to achieve the same. In the original formulation, the support objects of the LPSD tend to lie on the boundary, while for the LPDD, they are mostly 'inside' the boundary. This means that a removal of such an object will not impose a drastic change of the LPDD.

In summary, our LPDD method can be recommended when the failure to detect outliers is more expensive than the costs of a false alarm. It is also possible to enlarge the description of the LPDD by adding a small constant to $\rho$. Such a constant should be related to the dissimilarity values in the neighborhood of the boundary. How to choose it, remains an open issue for further research.

**Acknowledgements**. This work is partly supported by the Dutch Organization for Scientific Research (NWO) and the European Community Marie Curie Fellowship. The authors are solely responsible for information communicated and the European Commission is not responsible for any views or results expressed.

## Footnotes

[1]the prism is bounded if $D$ is bounded

[2] $P$ is not expected to be parallel to the prism's bottom hyperplane

## References

[1] K.P. Bennett and O.L. Mangasarian. Combining support vector and mathematical programming methods for induction. In B. Schölkopf, C.J.C. Burges, and A.J. Smola, editors, *Advances in Kernel Methods, Support Vector Learning*, pages 307–326. MIT Press, Cambridge, MA, 1999.

[2] C. Berg, J.P.R. Christensen, and P. Ressel. *Harmonic Analysis on Semigroups*. Springer-Verlag, 1984.

[3] C. Campbell and K.P. Bennett. A linear programming approach to novelty detection. In *Neural Information Processing Systems*, pages 395–401, 2000.

[4] M.P. Dubuisson and A.K. Jain. Modified Hausdorff distance for object matching. In *12th Internat. Conference on Pattern Recognition*, volume 1, pages 566–568, 1994.

[5] R.P.W. Duin. Compactness and complexity of pattern recognition problems. In *Internat. Symposium on Pattern Recognition 'In Memoriam Pierre Devijver'*, pages 124–128, Royal Military Academy, Brussels, 1999.

[6] R.P.W. Duin and E. Pękalska. Complexity of dissimilarity based pattern classes. In *Scandinavian Conference on Image Analysis*, 2001.

[7] D.W. Jacobs, D. Weinshall, and Y. Gdalyahu. Classification with non-metric distances: Image retrieval and class representation. *IEEE Trans. on PAMI*, 22(6):583–600, 2000.

[8] A.K. Jain and D. Zongker. Representation and recognition of handwritten digits using deformable templates. *IEEE Trans. on PAMI*, 19(12):1386–1391, 1997.

[9] Mangasarian O.L. Arbitrary-norm separating plane. *Operations Research Letters*, 24(1-2):15–23, 1999.

[10] E. Pękalska, P. Paclik, and R.P.W. Duin. A generalized kernel approach to dissimilarity-based classification. *Journal of Machine Learning Research*, 2(2):175–211, 2001.

[11] B. Schölkopf, J.C. Platt, A.J. Smola, and R.C. Williamson. Estimating the support of a high-dimensional distribution. *Neural Computation*, 13:1443–1471, 2001.

[12] B. Schölkopf, Williamson R.C., A.J. Smola, J. Shawe-Taylor, and J.C. Platt. Support vector method for novelty detection. In *Neural Information Processing Systems*, 2000.

[13] D.M.J. Tax. *One-class classification*. PhD thesis, Delft University of Technology, The Netherlands, 2001.

[14] D.M.J. Tax and R.P.W. Duin. Support vector data description. *Machine Learning*, 2002. accepted.

[15] V. Vapnik. *The Nature of Statistical Learning*. Springer, N.Y., 1995.

[16] http://www.sidanet.org.
